# Beyond Pair-Based STDP: a Phenomenogical Rule for Spike Triplet and Frequency Effects

**Jean-Pascal Pfister and Wulfram Gerstner**
School of Computer and Communication Sciences
and Brain-Mind Institute,
Ecole Polytechnique Fédérale de Lausanne (EPFL), CH-1015 Lausanne
{jean-pascal.pfister, wulfram.gerstner}@epfl.ch

## Abstract

While classical experiments on spike-timing dependent plasticity analyzed synaptic changes as a function of the timing of *pairs* of pre- and postsynaptic spikes, more recent experiments also point to the effect of spike *triplets*. Here we develop a mathematical framework that allows us to characterize timing based learning rules. Moreover, we identify a candidate learning rule with five variables (and 5 free parameters) that captures a variety of experimental data, including the dependence of potentiation and depression upon pre- and postsynaptic firing frequencies. The relation to the Bienenstock-Cooper-Munro rule as well as to some timing-based rules is discussed.

## 1 Introduction

Most experimental studies of Spike-Timing Dependent Plasticity (STDP) have focused on the timing of spike pairs [1, 2, 3] and so do many theoretical models. The spike-pair based models can be divided into two classes: either all pairs of spikes contribute in a homogeneous fashion [4, 5, 6, 7, 8, 9, 10] (called 'all-to-all' interaction in the following) or only pairs of 'neighboring' spikes [11, 12, 13] (called 'nearest-spike' interaction in the following); cf. [14, 15]. Apart from these phenomenological models, there are also models that are somewhat closer to the biophysics of synaptic changes [16, 17, 18, 19].

Recent experiments have furthered our understanding of timing effects in plasticity and added at least two different aspects: firstly, it has been shown that the mechanism of potentiation in STDP is different from that of depression [20] and secondly, it became clear that not only the timing of pairs, but also of triplets of spikes contributes to the outcome of plasticity experiments [21, 22].

In this paper, we introduce a learning rule that takes these two aspects partially into account in a simple way. Depression is triggered by *pairs* of spikes with *post-before-pre* timing, whereas potentiation is triggered by *triplets* of spikes consisting of 1 pre- and 2 postsynaptic spikes. Moreover, in our model the pair-based depression includes an explicit dependence upon the mean postsynaptic firing rate. We show that such a learning rule accounts for two important stimulation paradigms:

**P1 (Relative Spike Timing)**: *Both the pre- and postsynaptic spike trains consist of a burst*

*of N spikes at regular intervals T, but the two spike trains are shifted by a time $\Delta t = t^{\mathrm{post}} - t^{\mathrm{pre}}$.*

The total weight change is a function of the relative timing $\Delta t$ (this gives the standard STDP function), but also a function of the firing frequency $\rho = 1/T$ during the burst; cf. Fig. 1A (data from L5 pyramidal neurons in visual cortex).

**P2 (Poisson Firing)**: *The pre- and postsynaptic spike trains are generated by two independent Poisson processes with rates $\rho_x$ and $\rho_y$ respectively.*

Protocol P2 has less experimental support but it helps to establish a relation to the Bienenstock-Cooper-Munro (BCM) model [23]. To see that relation, it is useful to plot the weight change as a function of the postsynaptic firing rate, i.e., $\Delta w \propto \phi(\rho_y)$ (cf. Fig 1B). Note that the function $\phi$ has only been measured indirectly in experiments [24, 25].

We emphasize that in the BCM model,

$$\Delta w = \rho_x \phi(\rho_y, \bar{\rho}_y) \tag{1}$$

the function $\phi$ depends not only on the current firing rate $\rho_y$, but also on the *mean* firing rate $\bar{\rho}_y$ averaged over the recent past which has the effect that the threshold between depression and potentiation is not fixed but dynamic. More precisely, this threshold $\theta$ depends nonlinearly on the mean firing rate $\bar{\rho}_y$:

$$\theta = \alpha \bar{\rho}_y^p, \quad p > 1 \tag{2}$$

with parameters $\alpha$ and $p$. Previous models of STDP have already discussed the relation of STDP to the BCM rule [16, 12, 17, 26], but none of these seems to be completely satisfactory as discussed in Section 4. We will also compare our results to the rule of [21] which was together with the work of [16] amongst the first triplet rules to be proposed.

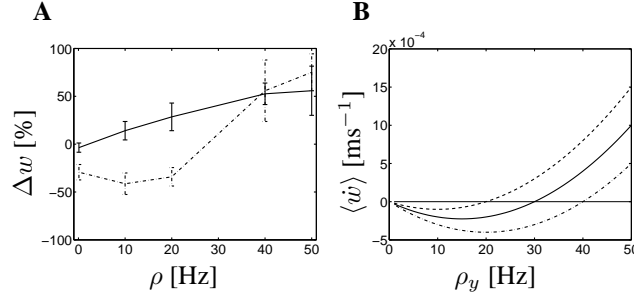

Figure 1: **A**. Weight change in an experiment on cortical synapses using pairing protocol (P1) (solid line: $\Delta t = 10$ ms, dot-dashed line $\Delta t = -10$ ms) as a function of the frequency $\rho$. Figure redrawn from [11]. **B**. Weight change in protocol P2 according to the BCM rule for $\theta = 20, 30, 40$ Hz.

## 2  A Framework for STDP

Several learning rules in the modeling literature can be classified according to the two criteria introduced above: (i) all-to-all interaction vs. nearest spike interaction; (ii) pair-based vs. triplet based rules. Point (ii) can be elaborated further in the context of an expansion (pairs, triplets, quadruplets, ... of spikes) that we introduce now.

### 2.1  Volterra Expansion ('all-to-all')

For the sake of simplicity, we assume that weight changes occur at the moment of presynaptic spike arrival or at the moment of postsynaptic firing. The direction and amplitude

of the weight change depends on the configuration of spikes in the presynaptic spike train $X(t) = \sum_k \delta(t - t_x^k)$ and the postsynaptic spike train $Y(t) = \sum_k \delta(t - t_y^k)$. With some arbitrary functionals $F[X, Y]$ and $G[X, Y]$, we write (see also [8])

$$\dot{w}(t) = X(t)F[X, Y] + Y(t)G[X, Y] \tag{3}$$

Clearly, there can be other neuronal variables that influence the synaptic dynamics. For example, the weight change can depend on the current weight value $w$ [8, 15, 10], the $Ca^{2+}$ concentration [17, 19], the depolarization [25, 27, 28], the mean postsynaptic firing rate $\bar{\rho}_y(t)$ [23],.... Here, we will consider only the dependence upon the history of the pre- and postsynaptic firing times and the mean postsynaptic firing rate $\bar{\rho}_y$. Note that even if $\bar{\rho}_y$ depends via a low-pass filter $\tau_\rho \dot{\bar{\rho}}_y = -\bar{\rho}_y + Y(t)$ on the past spike train $Y$ of the postsynaptic neuron, the description of the problem will turn out to be simpler if the mean firing rate is considered as a separate variable. Therefore, let us write the instantaneous weight change as

$$\dot{w}(t) = X(t)F([X, Y], \bar{\rho}_y(t)) + Y(t)G([X, Y], \bar{\rho}_y(t)) \tag{4}$$

The goal is now to determine the simplest functionals $F$ and $G$ that would be consistent with the experimental protocols $P1$ and $P2$ introduced above. Since the functionals are unknown, we perform a Volterra expansion of $F$ and $G$ in the hope that a small number of low-order terms are sufficient to explain a large body of experimental data. The Volterra expansion [29] of the functional $G$ can be written as[1]

$$
\begin{aligned}
G([X, Y]) &= G_1^y + \int_0^\infty G_2^{xy}(s)X(t - s)ds + \int_0^\infty G_2^{yy}(s)Y(t - s)ds \\
&+ \int_0^\infty \int_0^\infty G_3^{xxy}(s, s')X(t - s)X(t - s')ds'ds \\
&+ \int_0^\infty \int_0^\infty \mathbf{G}_3^{xyy}(s, s')X(t - s)Y(t - s')ds'ds \\
&+ \int_0^\infty \int_0^\infty G_3^{yyy}(s, s')Y(t - s)Y(t - s')ds'ds + \ldots \tag{5}
\end{aligned}
$$

Similarly, the expansion of $F$ yields

$$F([X, Y]) = F_1^x + \int_0^\infty F_2^{xx}(s)X(t - s)ds + \int_0^\infty \mathbf{F}_2^{xy}(s)Y(t - s)ds + \ldots \tag{6}$$

Note that the upper index in functions represents the type of interaction. For example, $G_3^{xyy}$ (in bold face above) refers to a triplet interaction consisting of 1 pre- and 2 postsynaptic spikes. Note that the $G_3^{xyy}$ term could correspond to a *pre-post-post* sequence as well as a *post-pre-post* sequence. Similarly the term $F_2^{xy}$ picks up the changes caused by arrival of a presynaptic spike after postsynaptic spike firing. Several learning rules with all-to-all interaction can be classified in this framework, e.g. [5, 6, 7, 8, 9, 10].

## 2.2 Our Model

Not all term in the expansion need to be non-zero. In fact, in the results section we will show that a learning rule with $G_3^{xyy}(s, s') \geq 0$ for all $s, s' > 0$ and $F_2^{xy}(s) \leq 0$ for $s > 0$ and all other terms set to zero is sufficient to explain the results from protocols P1 and P2. Thus, in our learning rule an isolated pair of spikes in configuration *post-before-pre* will lead to depression. An isolated spike pair *pre-before-post*, on the other hand, would not be sufficient to trigger potentiation, whereas a triplet *pre-post-post* or *post-pre-post* will do so (see Fig. 2).

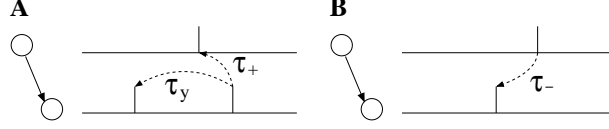

Figure 2: **A**. Triplet interaction for LTP **B**. Pair interaction for LTD.

To be specific, we consider

$$F_2^{xy}(s) = -A_-(\bar{\rho}_y)e^{-\frac{s}{\tau_-}} \quad \text{and} \quad G_3^{xyy}(s,s') = A_+ e^{-\frac{s}{\tau_+}} e^{-\frac{s'}{\tau_y}} . \tag{7}$$

Such an exponential model can be implemented by a mechanistic update involving three variables (the dot denotes a temporal derivative)

$$\dot{a} = -\frac{a}{\tau_+}; \quad \text{if } t = t_x^k \text{ then } a \to a + 1$$

$$\dot{b} = -\frac{b}{\tau_-}; \quad \text{if } t = t_y^k \text{ then } b \to b + 1 \tag{8}$$

$$\dot{c} = -\frac{c}{\tau_y}; \quad \text{if } t = t_y^k \text{ then } c \to c + 1$$

The weight update is then

$$\dot{w}(t) = -A_-(\bar{\rho}_y)X(t)b(t) + A_+Y(t)a(t)c(t). \tag{9}$$

### 2.3 Nearest Spike Expansion (truncated model)

Following ideas of [11, 12, 13], the expansion can also be restricted to neighboring spikes only. Let us denote by $f_y(t)$ the firing time of the last postsynaptic spike before time $t$. Similarly, $f_x(t')$ denotes the timing of the last presynaptic spike preceding $t'$. With this notation the Volterra expansion of the preceding section can be repeated in a form that only nearest spikes play a role. A classification of the models [11, 12, 13] is hence possible.

We focus immediately on the truncated version of our model

$$\dot{w}(t) = X(t)F_2^{xy}(t - f_y(t), \bar{\rho}_y(t)) + Y(t)G_3^{xyy}(t - f_x(t), t - f_y(t)) \tag{10}$$

The mechanistic model that generates the truncated version of the model is similar to Eq. (8) except that under the appropriate update condition, the variable goes to one, i.e. $a \to 1, b \to 1$ and $c \to 1$. The weight update is identical to that of the all-to-all model, Eq. (9).

## 3 Results

One advantage of our formulation is that we can derive explicit formulas for the total weight changes induced by protocols P1 and P2.

### 3.1 All-to-all Interaction

If we use protocol P1 with a total of $N$ pre- and postsynaptic spikes at frequency $\rho$ shifted by a time $\Delta t$, then the total weight change $\Delta w$ is for our model with all-to-all interaction

$$\begin{aligned}
\Delta w &= A_+ \sum_{k=0}^{N-1} \sum_{k'=1}^{N-1} (N - \max(k,k')) \exp\left(-\frac{k/\rho + \Delta t}{\tau_+}\right) \exp\left(-\frac{k'}{\tau_y\rho}\right) \lambda_k(-\Delta t) \\
&\quad - A_-(\bar{\rho}_y) \sum_{k=0}^{N-1} (N - k) \exp\left(-\frac{k/\rho - \Delta t}{\tau_-}\right) \lambda_k(\Delta t)
\end{aligned} \tag{11}$$

where $\lambda_k(\Delta t) = 1 - \delta_{k0}\Theta(\Delta t)$ with $\Theta$ the Heaviside step function. The results are plotted in Fig. 3 top-left for $N = 60$ spikes.

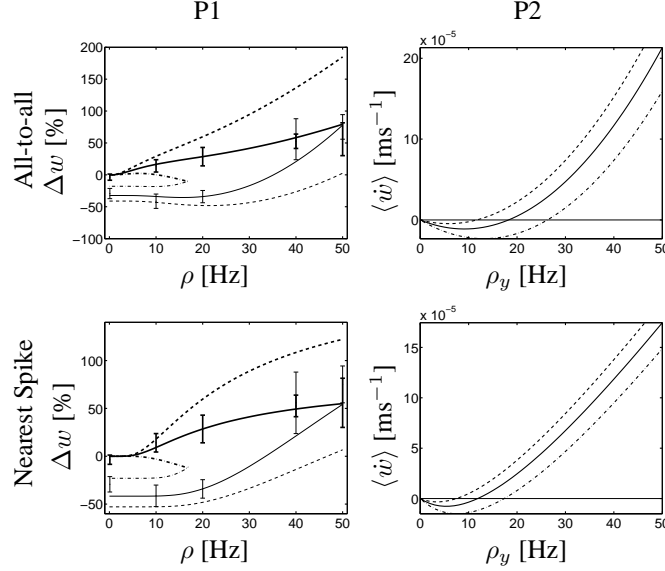

Figure 3: Triplet learning rule. Summary of all results of protocol $P1$ (left) and $P2$ (right) for an all-to-all (top) and nearest-spike (bottom) interaction scheme. For the left column, the upper thick lines correspond to positive timing ($\Delta t > 0$) while the lower thin lines to negative timing. Dashed line: $\Delta t = \pm 2$ ms, solid line: $\Delta t = \pm 10$ ms and dot-dashed line $\Delta t = \pm 30$ ms. The error bars indicate the experimental data points of Fig. 1A. Right column: dashed-line $\bar{\rho}_y = 8$ Hz, solid line $\bar{\rho}_y = 10$ Hz and dot-dashed line $\bar{\rho}_y = 12$ Hz. Top: $\tau_y = 200$ ms, bottom: $\tau_y = 40$ ms.

The mean firing rate $\bar{\rho}_y$ reflects the firing activity during the recent past (i.e. *before* the start of the experiment) and is assumed as fixed during the experiment. The exact value does not matter. Overall, the frequency dependence of changes $\Delta w$ is very similar to that observed in experiments. If $X$ and $Y$ are independent Poisson process, the protocol P2 gives a total weight change that can be calculated using standard arguments [8]

$$\langle \dot{w} \rangle = -A_-(\bar{\rho}_y)\rho_x\rho_y\tau_- + A_+\rho_x\rho_y^2\tau_+\tau_y \tag{12}$$

As before, the mean firing rate $\bar{\rho}_y$ reflects the firing activity during the recent past and is assumed as fixed during the experiment. In order to implement a sliding threshold as in the BCM rule, we take $A_-(\bar{\rho}_y) = \beta_-\bar{\rho}_y^2/\rho_0^2$ where we set $\rho_0 = 10$ Hz. This yields a frequency dependent threshold $\theta(\bar{\rho}_y) = \beta_-\tau_-\bar{\rho}_y^2/(A_+\tau_+\tau_y\rho_0^2)$. As can be seen in Fig. 3 top-right our model exhibits all essential features of a BCM rule.

### 3.2 Nearest Spike Interaction

We now apply protocols P1 and P2 to our truncated rule, i.e. restricted to the *nearest-spike* interaction; cf. Eq. (10) where the expression of $F_2^{xy}$ and $G_3^{xyy}$ are taken from Eq. (7). The weight change $\Delta w$ for the protocol $P1$ can be calculated explicitly and is plotted in Fig. 3 bottom-left. For protocol $P2$ (see Fig. 3 bottom-right) we find

$$\langle \dot{w} \rangle = \rho_x \left( -\frac{A_-(\bar{\rho}_y)\rho_y}{\rho_y + \alpha_-} + \frac{A_+}{\rho_x + \alpha_+}\frac{\rho_y^2}{\rho_y + \alpha_y} \right) \tag{13}$$

where $\alpha_y = \tau_y^{-1}$. If we assume that $\rho_x \ll \alpha_x$, Eq. (13) is a BCM learning rule.

In summary, both versions of our learning rule (all-to-all or nearest-spike) yield a frequency dependence that is consistent with experimental results under protocol P1 and with the BCM rule tested under protocol P2. We note that our learning rule contains only two terms, i.e., a triplet term (1 pre and 2 post) for potentiation and a *post-pre* pair term for depression. The dynamics is formulated using five variables $(a, b, c, \bar{\rho}_y, w)$ and five parameters $(\tau_+, \tau_-, \tau_y, A_+, \beta_-)$. $\tau_+ = 16.8$ ms and $\tau_- = 33.7$ ms are taken from [14]. $A_+$ and $\beta_-$ are chosen such that the weight changes for $\Delta t = \pm 10$ ms and $\rho = 20$ Hz fit the experimental data [11].

## 4 Discussion - Comparison with Other Rules

While we started out developing a general framework, we focused in the end on a simple model with only five parameters - why, then, this model and not some other combination of terms? To answer this question we apply our approach to a couple of other models, i.e., pair-based models (all-to-all or nearest spike), triplet-based models, and others.

### 4.1 STDP Models Based on Spike Pairs

Pair-based models with all-to-all interaction [4, 5, 6, 7, 8, 9, 10] yield under Poisson stimulation (protocol P2) a total weight change that is linear in presynaptic and postsynaptic frequencies. Thus, as a function of postsynaptic frequency we always find a straight line with a slope that depends on the integral of the STDP function [5, 7]. Thus pair-based models with all-to-all interaction need to be excluded in view of BCM features of plasticity [25, 24].

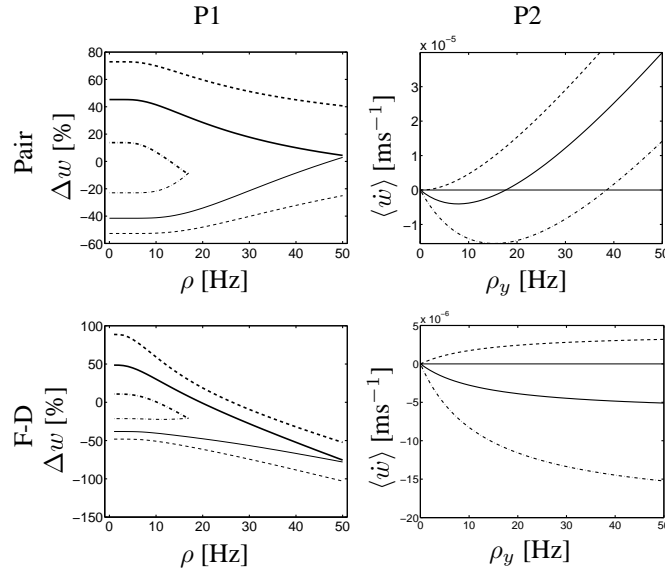

Figure 4: Pair learning rule in a nearest spike interaction scheme (top) and Froemke-Dan rule (bottom). For the left column, the higher thick lines correspond to positive timing ($\Delta t > 0$) while the lower thin lines to negative timing. Dashed line: $\Delta t = \pm 2$ ms, solid line: $\Delta t = \pm 10$ ms and dot-dashed line $\Delta t = \pm 30$ ms. Right column: dashed-line $\bar{\rho}_y = 8$ Hz, solid line $\bar{\rho}_y = 10$ Hz and dot-dashed line $\bar{\rho}_y = 12$ Hz. The parameters of the F-D model are taken from [21]. The dependence upon $\bar{\rho}_y$ has been added to the original F-D rule ($A_- \rightarrow \beta_- \bar{\rho}_y^2 / \rho_0^2$).

A pair-based model with nearest-spike interaction, however, can give a non-linear dependence upon the postsynaptic frequency under protocol P2 with fixed threshold between

depression and potentation [12]. We can go beyond the results of [12] by adding a suitable dependence of the parameter $A_-$ upon $\bar{\rho}_y$ which yields a sliding threshold; cf. Fig. 4 top right.

But even a pair rule restricted to nearest-spike interaction is unable to account for the results of protocol P1. An important feature of the experimental results with protocol P1 is that potentiation only occurs above a minimal firing frequency of the postsynaptic neuron (cf. Fig. 1A) whereas pair-based rules *always* exhibit potentiation with pre-before-post timing even in the limit of low frequencies; cf. Fig. 4 top left. The intuitive reason is that at low frequency the total weight change is proportional to the number of *pre-post* pairings and this argument can be directly transformed into a mathematical proof (details omitted). Thus, pair-based rules of potentiation (all-to-all or nearest spike) cannot account for results of protocol P1 and must be excluded.

### 4.2 Comparison with Triplet-Based Learning Rules

The model of Senn et al. [16] can well account of the results under protocol P1. A classification of this rule within our framework reveals that the update algorithm generates pair terms of the form *pre-post* and *post-pre*, as well as triplet terms of the form *pre-post-post* and *post-pre-pre*. As explained in the previous paragraph, a pair term *pre-post* generated potentiation even at very low frequencies which is not realistic. In order to avoid this effect in their model, Senn et al. included additional threshold values which increased the number of parameters in their model to 9 [16] while the number of variables is 5 as in our model. Moreover, the mapping of the model of Senn et al. to the BCM rule is not ideal, since the sliding threshold is different for each individual synapse [16].

An explicit triplet rule has been proposed by Froemke and Dan [21]. In our framework, the rule can be classified as a combination of triplet terms for potentiation and depression. Following the same line or argument as in the preceding sections we can calculate the total weight change for protocols P1 and P2. The result is shown in Fig. 4 bottom. We can clearly see that the pairing experiment $P1$ yields a behavior opposite to the one found experimentally and the BCM behavior is not at all reproduced in protocol P2.

### 4.3 Summary

We consider our model as a minimal model to account for results of protocol P1 and P2, but, of course, several factors are not captured by the model. First, our model has no dependence upon the current weight value, but, in principle, this could be included along the lines of [10]. Second, the model has no explicit dependence upon the membrane potential or calcium concentration, but the postsynaptic neuron enters only via its firing activity. Third, and most importantly, there are other experimental paradigms that have to be taken care of.

In a recent series of experiments Bi and colleagues [22] have systematically studied the effect of symmetric spike triplets (*pre-post-pre* or *post-pre-post*) and spike quadruplets (e.g., *pre-post-post-pre*) in hippocampal cultures. While the model presented in this paper is intended to model the synaptic dynamic for L5 pyramidal neurons in the visual cortex [11], it is possible to consider a similar model for the hippocampus containing two extra terms (a pair term for potentiation and and triplet term for depression).

## Footnotes

[1]For the sake of clarity we have omitted the dependence on $\bar{\rho}_y$.

## References

[1] Markram, H., Lübke, J., Frotscher, M., and Sakmann, B. *Science* **275**, 213–215 (1997).

[2] Zhang, L., Tao, H., Holt, C., W.A.Harris, and Poo, M.-M. *Nature* **395**, 37–44 (1998).

[3] Bi, G. and Poo, M. *Ann. Rev. Neurosci.* **24**, 139–166 (2001).

[4] Gerstner, W., Kempter, R., van Hemmen, J. L., and Wagner, H. *Nature* **383**, 76–78 (1996).

[5] Kempter, R., Gerstner, W., and van Hemmen, J. L. *Phys. Rev. E* **59**, 4498–4514 (1999).

[6] Roberts, P. *J. Computational Neuroscience* **7**, 235–246 (1999).

[7] Song, S., Miller, K., and Abbott, L. *Nature Neuroscience* **3**, 919–926 (2000).

[8] Kistler, W. M. and van Hemmen, J. L. *Neural Comput.* **12**, 385–405 (2000).

[9] Rubin, J., Lee, D. D., and Sompolinsky, H. *Physical Review Letters* **86**, 364–367 (2001).

[10] Gütig, R., Aharonov, R., Rotter, S., and Sompolinsky, H. *J. Neuroscience* **23**, 3697–3714 (2003).

[11] Sjöström, P., Turrigiano, G., and Nelson, S. *Neuron* **32**, 1149–1164 (2001).

[12] Izhikevich, E. and Desai, N. *Neural Computation* **15**, 1511–1523 (2003).

[13] Burkitt, A. N., Meffin, M. H., and Grayden, D. *Neural Computation* **16**, 885–940 (2004).

[14] Bi, G.-Q. *Biological Cybernetics* **319-332** (2002).

[15] van Rossum, M. C. W., Bi, G. Q., and Turrigiano, G. G. *J. Neuroscience* **20**, 8812–8821 (2000).

[16] Senn, W., Tsodyks, M., and Markram, H. *Neural Computation* **13**, 35–67 (2001).

[17] Shouval, H. Z., Bear, M. F., and Cooper, L. N. *Proc. Natl. Acad. Sci. USA* **99**, 10831–10836 (2002).

[18] Abarbanel, H., Huerta, R., and Rabinovich, M. *Proc. Natl. Academy of Sci. USA* **59**, 10137–10143 (2002).

[19] Karmarkar, U., Najarian, M., and Buonomano, D. *Biol. Cybernetics* **87**, 373–382 (2002).

[20] Sjöström, P., Turrigiano, G., and Nelson, S. *Neuron* **39**, 641–654 (2003).

[21] Froemke, R. and Dan, Y. *Nature* **416**, 433–438 (2002).

[22] Wang, H. X., Gerkin, R. C., Nauen, D. W., and Bi, G. Q. *Nature Neuroscience* **8**, 187–193 (2005).

[23] Bienenstock, E., Cooper, L., and Munro, P. *Journal of Neuroscience* **2**, 32–48 (1982). reprinted in Anderson and Rosenfeld, 1990.

[24] Kirkwood, A., Rioult, M. G., and Bear, M. F. *Nature* **381**, 526–528 (1996).

[25] Artola, A. and Singer, W. *Trends Neurosci.* **16**(11), 480–487 (1993).

[26] Toyoizumi, T., Pfister, J.-P., Aihara, K., and Gerstner, W. In *Advances in Neural Information Processing Systems 17,* Saul, L. K., Weiss, Y., and Bottou, L., editors, 1409–1416. MIT Press, Cambridge, MA (2005).

[27] Fusi, S., Annunziato, M., Badoni, D., Salamon, A., and D.J.Amit. *Neural Computation* **12**, 2227–2258 (2000).

[28] Toyoizumi, T., Pfister, J.-P., Aihara, K., and Gerstner, W. *Proc. National Academy Sciences (USA)* **102**, 5239–5244 (2005).

[29] Volterra, V. *Theory of Functionals and of Integral and Integro-Differential Equations*. Dover, New York, (1930).
